# Active Learning for Parameter Estimation in Bayesian Networks

**Simon Tong**
Computer Science Department
Stanford University
*simon.tong@cs.stanford.edu*

**Daphne Koller**
Computer Science Department
Stanford University
*koller@cs.stanford.edu*

## Abstract

*Bayesian networks* are graphical representations of probability distributions. In virtually all of the work on learning these networks, the assumption is that we are presented with a data set consisting of randomly generated instances from the underlying distribution. In many situations, however, we also have the option of *active learning*, where we have the possibility of guiding the sampling process by querying for certain types of samples. This paper addresses the problem of estimating the parameters of Bayesian networks in an active learning setting. We provide a theoretical framework for this problem, and an algorithm that chooses which active learning queries to generate based on the model learned so far. We present experimental results showing that our active learning algorithm can significantly reduce the need for training data in many situations.

## 1 Introduction

In many machine learning applications, the most time-consuming and costly task is the collection of a sufficiently large data set. Thus, it is important to find ways to minimize the number of instances required. One possible method for reducing the number of instances is to choose better instances from which to learn. Almost universally, the machine learning literature assumes that we are given a set of instances chosen randomly from the underlying distribution. In this paper, we assume that the learner has the ability to guide the instances it gets, selecting instances that are more likely to lead to more accurate models. This approach is called *active learning*.

The possibility of active learning can arise naturally in a variety of domains, in several variants. In *selective* active learning, we have the ability of explicitly asking for an example of a certain "type"; i.e., we can ask for an full instance where some of the attributes take on requested values. For example, if our domain involves webpages, the learner might be able to ask a human teacher for examples of homepages of graduate students in a Computer Science department. A variant of selective active learning is *pool-based* active learning, where the learner has access to a large pool of instances, about which it knows only the value of certain attributes. It can then ask for instances in this pool for which these known attributes take on certain values. For example, one could redesign the U.S. census to have everyone fill out only the short form; the active learner could then select among the respondents for those that should fill out the more detailed long form. Another example is a cancer study in which we have a list of people's ages and whether they smoke, and we can ask a subset of these people to undergo a thorough examination.

In such active learning settings, we need a mechanism that tells us which instances to select. This problem has been explored in the context of supervised learning [1, 2, 7, 9]. In this paper, we consider its application in the unsupervised learning task of density estimation. We present a formal framework for active learning in *Bayesian networks (BNs)*.

We assume that the graphical structure of the BN is fixed, and focus on the task of parameter estimation. We define a notion of model accuracy, and provide an algorithm that selects queries in a greedy way, designed to improve model accuracy as much as possible. At first sight, the applicability of active learning to density estimation is unclear. Given that we are not simply sampling , it is initially not clear that an active learning algorithm even learns the correct density. In fact we can actually show that our algorithm is *consistent*, i.e., it converges to the right density at the limit. Furthermore, it is not clear that active learning is necessarily beneficial in this setting. After all, if we are trying to estimate a distribution, then random samples from that distribution would seem the best source. Surprisingly, we provide empirical evidence showing that, in a range of interesting circumstances, our approach learns from significantly fewer instances than random sampling.

## 2 Learning Bayesian Networks

Let $\mathcal{X} = \{X_1, \ldots, X_n\}$ be a set of random variables, with each variable $X_i$ taking values in some finite domain $Dom[X_i]$. A *Bayesian network* over $\mathcal{X}$ is a pair $(\mathcal{G}, \boldsymbol{\theta})$ that represents a distribution over the joint space of $\mathcal{X}$. $\mathcal{G}$ is a directed acyclic graph, whose nodes correspond to the random variables in $\mathcal{X}$ and whose structure encodes conditional independence properties about the joint distribution. We use $\mathbf{U}_i$ to denote the set of parents of $X_i$. $\boldsymbol{\theta}$ is a set of parameters which quantify the network by specifying the *conditional probability distributions (CPDs)* $P(X_i \mid \mathbf{U}_i)$. We assume that the CPD of each node consists of a separate multinomial distribution over $Dom[X_i]$ for each instantiation $\mathbf{u}$ of the parents $\mathbf{U}_i$. Hence, we have a parameter $\theta_{x_{ij}|\mathbf{u}}$ for each $x_{ij} \in Dom[X_i]$; we use $\boldsymbol{\theta}_{X_i|\mathbf{u}}$ to represent the vector of parameters associated with the multinomial $P(X_i \mid \mathbf{u})$.

Our focus is on the *parameter estimation* task: we are given the network structure $\mathcal{G}$, and our goal is to use data to estimate the network parameters $\boldsymbol{\theta}$. We will use Bayesian parameter estimation, keeping a density over possible parameter values. As usual [5], we make the assumption of *parameter independence*, which allows us to represent the joint distribution $p(\boldsymbol{\theta})$ as a set of independent distributions, one for each multinomial $\boldsymbol{\theta}_{X_i|\mathbf{u}}$.

For multinomials, the conjugate prior is a *Dirichlet* distribution [4], which is parameterized by *hyperparameters* $\alpha_j \in \mathbb{R}^+$, with $\alpha_* = \sum_j \alpha_j$. Intuitively, $\alpha_j$ represents the number of "imaginary samples" observed prior to observing any data. In particular, if $X$ is distributed multinomial with parameters $\boldsymbol{\theta} = (\theta_1, \ldots, \theta_r)$, and $p(\boldsymbol{\theta})$ is Dirichlet, then the probability that our next observation is $x_j$ is $\alpha_j/\alpha_*$. If we obtain a new instance $X = x_j$ sampled from this distribution, then our posterior distribution $p(\boldsymbol{\theta})$ is also distributed Dirichlet with hyperparameters $(\alpha_1, \ldots, \alpha_j + 1, \ldots, \alpha_r)$. In a BN with the parameter independence assumption, we have a Dirichlet distribution for every multinomial distribution $\boldsymbol{\theta}_{X_i|\mathbf{u}}$. Given a distribution $p(\boldsymbol{\theta})$, we use $\alpha_{x_{ij}|\mathbf{u}}$ to denote the hyperparameter corresponding to the parameter $\theta_{x_{ij}|\mathbf{u}}$.

## 3 Active Learning

Assume we start out with a network structure $\mathcal{G}$ and a prior distribution $p(\boldsymbol{\theta})$ over the parameters of $\mathcal{G}$. In a standard machine learning framework, data instances are independently, randomly sampled from some underlying distribution. In an active learning setting, we have the ability to request certain types of instances. We formalize this idea by assuming that some subset $\mathcal{C}$ of the variables are *controllable*. The learner can select a subset of variables $\mathbf{Q} \subset \mathcal{C}$ and a particular instantiation $\mathbf{q}$ to $\mathbf{Q}$. The request $\mathbf{Q} = \mathbf{q}$ is called a *query*. The result of such a query is a randomly sampled instance $\mathbf{x}$ conditioned on $\mathbf{Q} = \mathbf{q}$.

A *(myopic) active learner* $\ell$ is a querying function that takes $\mathcal{G}$ and $p(\boldsymbol{\theta})$, and selects a query $\mathbf{Q} = \mathbf{q}$. It takes the resulting instance $\mathbf{x}$, and uses it to update its distribution $p(\boldsymbol{\theta})$ to obtain a posterior $p'(\boldsymbol{\theta})$. It then repeats the process, using $p'$ for $p$. We note that $p(\boldsymbol{\theta})$ summarizes all the relevant aspects of the data seen so far, so that we do not need to maintain the history of previous instances. To fully specify the algorithm, we need to address two issues: we need to describe how our parameter distribution is updated given

that **x** is not a random sample, and we need to construct a mechanism for selecting the next query based on $p$.

To answer the first issue assume for simplicity that our query is $Q = q$ for a single node $Q$. First, it is clear that we cannot use the resulting instance **x** to update the parameters of the node $Q$ itself. However, we also have a more subtle problem. Consider a parent $U$ of $Q$. Although **x** does give us information about the distribution of $U$, it is not information that we can conveniently use. Intuitively, $P(U \mid Q = q)$ is sampled from a distribution specified by a complex formula involving multiple parameters. We avoid this problem simply by ignoring the information provided by **x** on nodes that are "upstream" of $Q$. More generally, we define a variable $Y$ to be *updateable in the context of a selective query* **Q** if it is not in **Q** or an ancestor of a node in **Q**.

Our update rule is now very simple. Given a prior distribution $p(\boldsymbol{\theta})$ and an instance **x** from a query **Q** = **q**, we do standard Bayesian updating, as in the case of randomly sampled instances, but we update only the Dirichlet distributions of updateable nodes. We use $p(\boldsymbol{\theta} \dagger \mathbf{Q} = \mathbf{q}, \mathbf{x})$ to denote the distribution $p'(\boldsymbol{\theta})$ obtained from this algorithm; this can be read as "the density of $\boldsymbol{\theta}$ after asking query **q** and obtaining the response **x**".

Our second task is to construct an algorithm for deciding on our next query given our current distribution $p$. The key step in our approach is the definition of a measure for the quality of our learned model. This allows us to evaluate the extent to which various instances would improve the quality of our model, thereby providing us with an approach for selecting the next query to perform. Our formulation is based on the framework of *Bayesian point estimation*. In the Bayesian learning framework, we maintain a distribution $p(\boldsymbol{\theta})$ over all of the model parameters. However, when we are asked to reason using the model, we typically "collapse" this distribution over parameters, generate a single representative model $\tilde{\boldsymbol{\theta}}$, and answer questions relative to that. If we choose to use $\tilde{\boldsymbol{\theta}}$, whereas the "true" model is $\boldsymbol{\theta}^*$, we incur some loss $Loss(\tilde{\boldsymbol{\theta}} \parallel \boldsymbol{\theta}^*)$. Our goal is to minimize this loss. Of course, we do not have access to $\boldsymbol{\theta}^*$. However, our posterior distribution $p(\boldsymbol{\theta})$ represents our "optimal" beliefs about the different possible values of $\boldsymbol{\theta}^*$, given our prior knowledge and the evidence. Therefore, we can define the *risk* of a particular $\tilde{\boldsymbol{\theta}}$ with respect to $p$ as:

$$E_{\Theta \sim p(\boldsymbol{\theta})}[Loss(\Theta \parallel \tilde{\boldsymbol{\theta}})] = \int_{\theta} Loss(\boldsymbol{\theta} \parallel \tilde{\boldsymbol{\theta}})p(\boldsymbol{\theta}) \, d\boldsymbol{\theta}. \quad (1)$$

We then define the *Bayesian point estimate* to be the value of $\tilde{\boldsymbol{\theta}}$ that minimizes the risk. We shall only be considering using the Bayesian point estimate, thus we define the *risk of a density p*, $Risk(p(\boldsymbol{\theta}))$, to be the risk of the optimal $\tilde{\boldsymbol{\theta}}$ with respect to $p$.

The risk of our density $p(\boldsymbol{\theta})$ is our measure for the quality of our current state of knowledge, as represented by $p(\boldsymbol{\theta})$. In a greedy scheme, our goal is to obtain an instance **x** such that the risk of the $p'$ obtained by updating $p$ with **x** is lowest. Of course, we do not know exactly which **x** we are going to get. We know only that it will be sampled from a distribution induced by our query. Our *expected posterior risk* is therefore:

$$ExPRisk(p(\boldsymbol{\theta}) \mid \mathbf{Q} = \mathbf{q}) = E_{\Theta \sim p(\boldsymbol{\theta})} E_{\mathbf{x} \sim P_{\Theta}(\mathbf{X}|\mathbf{Q}=\mathbf{q})} Risk(p(\boldsymbol{\theta} \dagger \mathbf{Q} = \mathbf{q}, \mathbf{x})). \quad (2)$$

This definition leads immediately to the following simple algorithm: For each candidate query **Q** = **q**, we evaluate the expected posterior risk, and then select the query for which it is lowest.

## 4   Active Learning Algorithm

To obtain a concrete algorithm from the active learning framework shown in the previous section, we must pick a loss function. There are many possible choices, but perhaps the best justified is the *relative entropy* or *Kullback-Leibler divergence (KL-divergence)* [3]: $KL(\boldsymbol{\theta} \parallel \tilde{\boldsymbol{\theta}}) = \sum_{\mathbf{x}} P_{\boldsymbol{\theta}}(\mathbf{x}) \ln \frac{P_{\boldsymbol{\theta}}(\mathbf{x})}{P_{\tilde{\boldsymbol{\theta}}}(\mathbf{x})}$. The KL-divergence has several independent justifications, and a variety of properties that make it particularly suitable as a measure of distance between distributions. We therefore proceed in this paper using KL-divergence as our

loss function. (An analogous analysis can be carried through for another very natural loss function: negative loglikelihood of future data — in the case of multinomial CPDs with Dirichlet densities over the parameters this results in an identical final algorithm.)

We now want to find an efficient approach to computing the risk. Two properties of KL-divergence turn out to be crucial. The first is that the value $\tilde{\boldsymbol{\theta}}$ that minimizes the risk relative to $p$ is the mean value of the parameters, $E_{\Theta \sim p(\boldsymbol{\theta})}[\boldsymbol{\theta}]$. For a Bayesian network with independent Dirichlet distributions over the parameters, this expression reduces to $\tilde{\boldsymbol{\theta}}_{x_{ij}|\mathbf{u}} = \frac{\alpha_{x_{ij}|\mathbf{u}}}{\alpha_{x_{i*}|\mathbf{u}}}$, the standard (Bayesian) approach used for collapsing a distribution over BN models into a single model. The second observation is that, for BNs, KL-divergence decomposes with the graphical structure of the network:

$$KL(\boldsymbol{\theta} \parallel \boldsymbol{\theta}') = \sum_i KL(P_{\boldsymbol{\theta}}(X_i \mid \mathbf{U}_i) \parallel P_{\boldsymbol{\theta}'}(X_i \mid \mathbf{U}_i)), \tag{3}$$

where $KL(P(X_i \mid \mathbf{U}_i) \parallel P'(X_i \mid \mathbf{U}_i))$ is the *conditional KL-divergence* and is given by $\sum_{\mathbf{u}} P(\mathbf{u})KL(P(X_i \mid \mathbf{u}) \parallel P'(X_i \mid \mathbf{u}))$. With these two facts, we can prove the following:

**Theorem 4.1** *Let* $\Gamma(\alpha)$ *be the Gamma function,* $\Psi(\alpha)$ *be the* digamma *function* $\Gamma'(\alpha)/\Gamma(\alpha)$, *and* $H$ *be the entropy function. Define* $\delta(\alpha_1,\ldots,\alpha_r) = \sum_{j=1}^{r} \left[ \frac{\alpha_j}{\alpha_*} \left( \Psi(\alpha_j + 1) - \Psi(\alpha_* + 1) \right) + H\left( \frac{\alpha_1}{\alpha_*}, \ldots, \frac{\alpha_r}{\alpha_*} \right) \right]$. *Then the risk decomposes as:*

$$Risk(p(\boldsymbol{\theta})) = \sum_i \sum_{\mathbf{u} \in Dom[\mathbf{U}_i]} P_{\tilde{\boldsymbol{\theta}}}(\mathbf{u})\delta(\alpha_{x_{i1}|\mathbf{u}}, \ldots, \alpha_{x_{ir_i}|\mathbf{u}}). \tag{4}$$

Eq. (4) gives us a concrete expression for evaluating the risk of $p(\boldsymbol{\theta})$. However, to evaluate a potential query, we also need its expected posterior risk. Recall that this is the expectation, over all possible answers to the query, of the risk of the posterior distribution $p'$. In other words, it is an average over an exponentially large set of possibilities.

To understand how we can evaluate this expression efficiently, we first consider a much simpler case. Consider a BN where we have only one child node $X$ and its parents $\mathbf{U}$, i.e., the only edges are from the nodes $\mathbf{U}$ to $X$. We also restrict attention to queries where we control all and only the parents $\mathbf{U}$. In this case, a query $\mathbf{q}$ is an instantiation to $\mathbf{U}$, and the possible outcomes to the query are the possible values of the variable $X$.

The expected posterior risk contains a term for each variable $X_i$ and each instantiation to its parents. In particular, it contains a term for each of the parent variables $U$. However, as these variables are not updateable, their hyperparameters remain the same following any query $\mathbf{q}$. Hence, their contribution to the risk is the same in every $p(\boldsymbol{\theta} \dagger \mathbf{U} = \mathbf{q}, x)$, and in our prior $p(\boldsymbol{\theta})$. Thus, we can ignore the terms corresponding to the parents, and focus on the terms associated with the conditional distribution $P(X \mid \mathbf{U})$. Hence, we have:

$$Risk_X(p(\boldsymbol{\theta})) = \sum_{\mathbf{u}} P_{\tilde{\boldsymbol{\theta}}}(\mathbf{u})\delta(\alpha_{x_1|\mathbf{u}}, \ldots, \alpha_{x_r|\mathbf{u}}) \tag{5}$$

$$ExPRisk_X(p(\boldsymbol{\theta}) \mid \mathbf{U} = \mathbf{q}) = \sum_{j} P_{\tilde{\boldsymbol{\theta}}}(x_j \mid \mathbf{q}) \sum_{\mathbf{u}} P_{\tilde{\boldsymbol{\theta}}'}(\mathbf{u})\delta(\alpha'_{x_1|\mathbf{u}}, \ldots, \alpha'_{x_r|\mathbf{u}}), \tag{6}$$

where $\alpha'_{x_j|\mathbf{u}}$ is the hyperparameter in $p(\boldsymbol{\theta} \dagger \mathbf{Q} = \mathbf{q}, x_j)$.

Rather than evaluating the expected posterior risk directly, we will evaluate the reduction in risk obtained by asking a query $\mathbf{U} = \mathbf{q}$:

$$\Delta(X \mid \mathbf{q}) = Risk(p(\boldsymbol{\theta})) - ExPRisk(p(\boldsymbol{\theta}) \mid \mathbf{q}) = Risk_X(p(\boldsymbol{\theta})) - ExPRisk_X(p(\boldsymbol{\theta}) \mid \mathbf{q})$$

Our first key observation relies on the fact that the variables $\mathbf{U}$ are not updateable for this query, so that their hyperparameters do not change. Hence, $P_{\tilde{\boldsymbol{\theta}}}(\mathbf{u})$ and $P_{\tilde{\boldsymbol{\theta}}'}(\mathbf{u})$ are the same. The second observation is that the hyperparameters corresponding to an instantiation $\mathbf{u}$ are the same in $p$ and $p'$ except for $\mathbf{u} = \mathbf{q}$. Hence, terms cancel and the expression simplifies to: $P_{\tilde{\boldsymbol{\theta}}}(\mathbf{q}) \left( \delta(\alpha_{x_1|\mathbf{q}}, \ldots, \alpha_{x_r|\mathbf{q}}) - \sum_j P_{\tilde{\boldsymbol{\theta}}}(x_j \mid \mathbf{q})\delta(\alpha'_{x_1|\mathbf{q}}, \ldots, \alpha'_{x_r|\mathbf{q}}) \right)$. By taking advantage of certain functional properties of $\Psi$, we finally obtain:

$$\Delta(X \mid \mathbf{q}) = P_{\tilde{\boldsymbol{\theta}}}(\mathbf{q}) \left( H\left( \frac{\alpha_{x_1|\mathbf{q}}}{\alpha_{x_*|\mathbf{q}}}, \ldots, \frac{\alpha_{x_r|\mathbf{q}}}{\alpha_{x_*|\mathbf{q}}} \right) - \sum_j P_{\tilde{\boldsymbol{\theta}}}(x_j \mid \mathbf{q}) H\left( \frac{\alpha'_{x_1|\mathbf{q}}}{\alpha'_{x_*|\mathbf{q}}}, \ldots, \frac{\alpha'_{x_r|\mathbf{q}}}{\alpha'_{x_*|\mathbf{q}}} \right) \right) \tag{7}$$

If we now select our query $\mathbf{q}$ so as to maximize the difference between our current risk and the expected posterior risk, we get a very natural behavior: We will select the query $\mathbf{q}$ that leads to the greatest reduction in the entropy of $X$ given its parents. It is also here that we can gain an insight as to where active learning has an edge over random sampling. Consider one situation in which $\mathbf{q}_1$ which is 100 times less likely than $\mathbf{q}_2$; $\mathbf{q}_1$ will lead us to update a parameter whose current density is $Dirichlet(1,1)$, whereas $\mathbf{q}_2$ will lead us to update a parameter whose current density is $Dirichlet(100, 100)$. However, according to $\Delta$, updating the former is worth *more* than the latter. In other words, if we are confident about commonly occurring situations, it is worth more to ask about the rare cases.

We now generalize this derivation to the case of an arbitrary BN and an arbitrary query. Here, our average over possible query answers encompasses exponentially many terms. Fortunately, we can utilize the structure of the BN to avoid an exhaustive enumeration.

**Theorem 4.2** *For an arbitrary BN and an arbitrary query* $\mathbf{Q} = \mathbf{q}$, *the expected KL posterior risk decomposes as:*

$$ExPRisk(p(\boldsymbol{\theta}) \mid \mathbf{Q} = \mathbf{q}) = \sum_i \sum_{\mathbf{u} \in Dom[\mathbf{U}_i]} P_{\bar{\boldsymbol{\theta}}}(\mathbf{u} \mid \mathbf{Q} = \mathbf{q}) ExPRisk_{X_i}(p(\boldsymbol{\theta}) \mid \mathbf{U}_i = \mathbf{u}).$$

In other words, the expected posterior risk is a weighted sum of expected posterior risks for conditional distributions of individual nodes $X_i$, where for each node we consider "queries" that are complete instantiations to the parents $\mathbf{U}_i$ of $X_i$.

We now have similar decompositions for the risk and the expected posterior risk. The obvious next step is to consider the difference between them, and then simplify it as we did for the case of a single variable. Unfortunately, in the case of general BNs, we can no longer exploit one of our main simplifying assumptions. Recall that, in the expression for the risk (Eq. (5)), the term involving $X_i$ and $\mathbf{u}$ is weighted by $P_{\bar{\boldsymbol{\theta}}}(\mathbf{u})$. In the expected posterior risk, the weight is $P_{\bar{\boldsymbol{\theta}}'}(\mathbf{u})$. In the case of a single node and a full parent query, the hyperparameters of the parents could not change, so these two weights were necessarily the same. In the more general setting, an instantiation $\mathbf{x}$ can change hyperparameters all through the network, leading to different weights.

However, we believe that a single data instance will not usually lead to a dramatic change in the distributions. Hence, these weights are likely to be quite close. To simplify the formula (and the associated computation), we therefore choose to approximate the posterior probability $P_{\bar{\boldsymbol{\theta}}'}(\mathbf{u})$ using the prior probability $P_{\bar{\boldsymbol{\theta}}}(\mathbf{u})$. Under this assumption, we can use the same simplification as we did in the single node case.

Assuming that this approximation is a good one, we have that:

$$\Delta(\mathcal{X} \mid \mathbf{q}) = Risk(p(\boldsymbol{\theta})) - ExPRisk(p(\boldsymbol{\theta}) \mid \mathbf{q}) \approx \sum_i \sum_{\mathbf{u} \in Dom[\mathbf{U}_i]} P_{\bar{\boldsymbol{\theta}}}(\mathbf{u} \mid \mathbf{q})\Delta(X_i \mid \mathbf{u}),$$

$$(8)$$

where $\Delta(X_i \mid \mathbf{u})$ is as defined in Eq. (7). Notice that we actually only need to sum over the updateable $X_i$s since $\Delta(X_i \mid \mathbf{u})$ will be zero for all non-updateable $X_i$s.

The above analysis provides us with an efficient implementation of our general active learning scheme. We simply choose a set of variables in the Bayesian network that we wish to control, and for each instantiation of the controllable variables we compute the expected change in risk given by Eq. (8). We then ask the query with the greatest expected change and update the parameters of the updateable nodes.

We now consider the computational complexity of the algorithm. It turns out that, for each potential query, all of the desired quantities can be obtained via two inference passes using a standard join tree algorithm [6]. Thus, the run time complexity of the algorithm is: $\mathcal{O}(|\mathcal{Q}| \cdot \text{cost of BN join tree inference})$, where $\mathcal{Q}$ is the set of candidate queries.

Our algorithm (approximately) finds the query that reduces the expected risk the most. We can show that our specific querying scheme (including the approximation) is *consistent*. As we mentioned before, this statement is non-trivial and depends heavily on the specific querying algorithm.

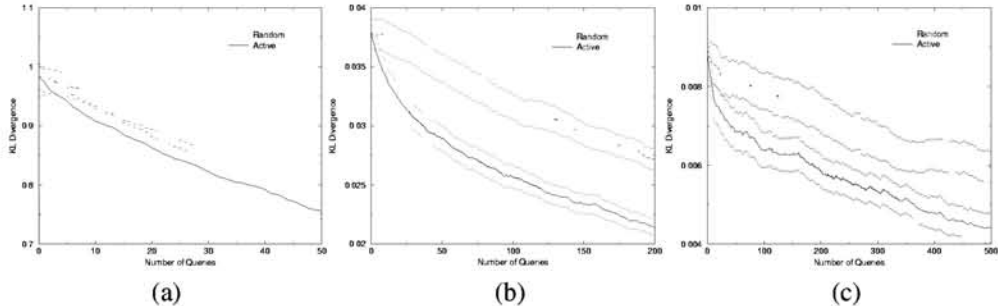

Figure 1: (a) **Alarm** network with three controllable nodes. (b) **Asia** network with two controllable nodes. (c) **Cancer** network with one controllable node. The axes are zoomed for resolution.

**Theorem 4.3** *Let $\mathcal{U}$ be the set of nodes which are updateable for at least one candidate query at each querying step. Assuming that the underlying true distribution is not deterministic, then our querying algorithm produces consistent estimates for the CPD parameters of every member of $\mathcal{U}$.*

## 5   Experimental Results

We performed experiments on three commonly used networks: **Alarm**, **Asia** and **Cancer**. **Alarm** has 37 nodes and 518 independent parameters, **Asia** has eight nodes and 18 independent parameters, and **Cancer** has five nodes and 11 independent parameters.

We first needed to set the priors for each network. We use the standard approach [5] of eliciting a network and an equivalent sample size. In our experiments, we assumed that we had fairly good background knowledge of the domain. To simulate this, we obtained our prior by sampling a few hundred instances from the true network and used the counts (together with smoothing from a uniform prior) as our prior. This is akin to asking for a prior network from a domain expert, or using an existing set of complete data to find initial settings of the parameters. We then compared refining the parameters either by using active learning or by random sampling. We permitted the active learner to abstain from choosing a value for a controlled node if it did not wish to — that node is then sampled as usual.

Figure 1 presents the results for the three networks. The graphs compare the KL-divergence between the learned networks and the true network that is generating the data. We see that active learning provides a substantial improvement in all three networks. The improvement in the **Alarm** network is particularly striking given that we had control of just three of the 36 nodes. The extent of the improvement depends on the extent to which queries allow us to reach rare events. For example, *Smoking* is one of the controllable variables in the **Asia** network. In the original network, $P(Smoking) = 0.5$. Although there was a significant gain by using active learning in this network, we found that there was a greater increase in performance if we altered the generating network to have $P(Smoking) = 0.9$; this is the graph that is shown.

We also experimented with specifying uniform priors with a small equivalent sample size. Here, we obtained significant benefit in the **Asia** network, and some marginal improvement in the other two. One possible reason is that the improvement is "washed out" by randomness, as the active learner and standard learner are learning from different instances. Another explanation is that the approximation in Eq. (8) may not hold as well when the prior $p(\boldsymbol{\theta})$ is uninformed and thereby easily perturbed even by a single instance. This indicates that our algorithm may perform best when refining an existing domain model.

Overall, we found that in almost all situations active learning performed as well as or better than random sampling. The situations where active learning produced most benefit were, unsurprisingly, those in which the prior was confident and correct about the commonly occurring cases and uncertain and incorrect about the rare ones. Clearly, this is the precisely the scenario we are most likely to encounter in practice when the prior is elicited

from an expert. By experimenting with forcing different priors we found that active learning was worse in one type of situation: where the prior was confident yet incorrect about the commonly occurring cases and uncertain but actually correct about the rare ones. This type of scenario is unlikely to occur in practice. Another factor affecting the performance was the degree to which the controllable nodes could influence the updateable nodes.

## 6   Discussion and Conclusions

We have presented a formal framework and resulting querying algorithm for parameter estimation in Bayesian networks. To our knowledge, this is one of the first applications of active learning in an unsupervised context. Our algorithm uses parameter distributions to guide the learner to ask queries that will improve the quality of its estimate the most.

BN active learning can also be performed in a *causal* setting. A query now acts as experiment – it intervenes in a model and forces variables to take particular values. Using Pearl's intervention theory [8], we can easily extend our analysis to deal with this case. The only difference is that the notion of an updateable node is even simpler — any node that is not part of a query is updateable. Regrettably, space prohibits a more complete exposition.

We have demonstrated that active learning can have significant advantages for the task of parameter estimation in BNs, particularly in the case where our parameter prior is of the type that a human expert is likely to provide. Intuitively, the benefit comes from estimating the parameters associated with rare events. Although it is less important to estimate the probabilities of rare events accurately, the number of instances obtained if we randomly sample from the distribution is still not enough. We note that this advantage arises even when we have used a loss function that considers only the accuracy of the distribution. In many practical settings such as medical or fault diagnosis, the rare cases are even more important, as they are often the ones that it is critical for the system to deal with correctly.

A further direction that we are pursuing is active learning for the *causal structure* of a domain. In other words, we are presented with a domain whose causal structure we wish to understand and we want to know the best sequence of experiments to perform.

**Acknowledgements**   The experiments were performed using the PHROG system, developed primarily by Lise Getoor, Uri Lerner, and Ben Taskar. Thanks to Carlos Guestrin and Andrew Ng for helpful discussions. The work was supported by DARPA's *Information Assurance* program under subcontract to SRI International, and by ARO grant DAAH04-96-1-0341 under the MURI program "Integrated Approach to Intelligent Systems".

## References

[1]   A.C. Atkinson and A.N. Donev. *Optimal Experimental Designs*. Oxford University Press, 1992.

[2]   D. Cohn, Z. Ghahramani, and M. Jordan. Active learning with statistical models. *Journal of Artificial Intelligence Research*, 4, 1996.

[3]   T.M Cover and J.A. Thomas. *information Theory*. Wiley, 1991.

[4]   M. H. DeGroot. *Optimal Statistical Decisions*. McGraw-Hill, New York, 1970.

[5]   D. Heckerman, D. Geiger, and D. M. Chickering. Learning Bayesian networks: The combination of knowledge and statistical data. *Machine Learning*, 20:197–243, 1995.

[6]   S. L. Lauritzen and D. J. Spiegelhalter. Local computations with probabilities on graphical structures and their application to expert systems. *J. Royal Statistical Society*, B 50(2), 1988.

[7]   D. MacKay. Information-based objective functions for active data selection. *Neural Computation*, 4:590–604, 1992.

[8]   J. Pearl. *Causality: Models, Reasoning, and Inference*. Cambridge University Press, 2000.

[9]   H.S. Seung, M. Opper, and H. Sompolinsky. Query by committee. In *Proc. COLT*, pages 287–294, 1992.